# Pose-Sensitive Embedding
# by Nonlinear NCA Regression

**Graham W. Taylor, Rob Fergus, George Williams, Ian Spiro and Christoph Bregler**
Courant Institute of Mathematics, New York University
New York, USA 10003
`gwtaylor,fergus,spiro,bregler@cs.nyu.edu`

## Abstract

This paper tackles the complex problem of visually matching people in similar pose but with different clothes, background, and other appearance changes. We achieve this with a novel method for learning a nonlinear embedding based on several extensions to the Neighborhood Component Analysis (NCA) framework. Our method is convolutional, enabling it to scale to realistically-sized images. By cheaply labeling the head and hands in large video databases through Amazon Mechanical Turk (a crowd-sourcing service), we can use the task of localizing the head and hands as a proxy for determining body pose. We apply our method to challenging real-world data and show that it can generalize beyond hand localization to infer a more general notion of body pose. We evaluate our method quantitatively against other embedding methods. We also demonstrate that real-world performance can be improved through the use of synthetic data.

## 1   Introduction

Determining the pose of a human body from one or more images is a central problem in Computer Vision. The complex, multi-jointed nature of the body makes the determination of pose challenging, particularly in natural settings where ambiguous and unusual configurations may be observed. The ability to localize the hands is particularly important: they provide tight constraints on the layout of the upper body, yielding a strong cue as to the action and intent of a person.

A huge range of techniques, both parametric and non-parametric, exist for inferring body pose from 2D images and 3D datasets [10, 39, 4, 28, 33, 8, 3, 6, 11]. We propose a non-parametric approach to

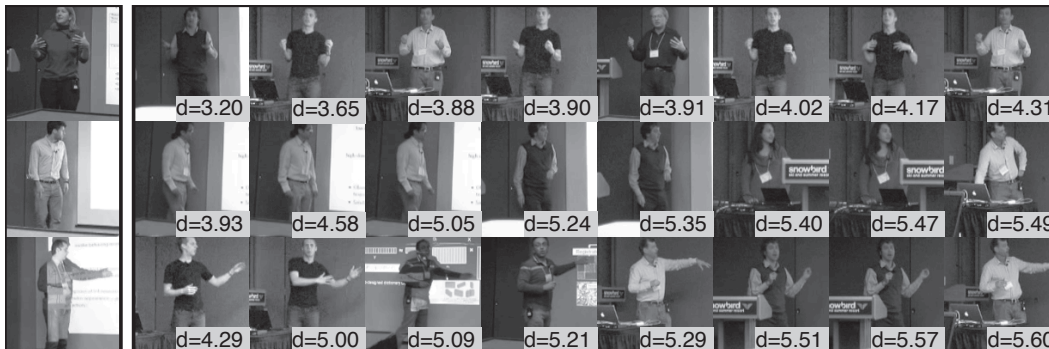

Figure 1: Query image (in left column) and the eight nearest neighbours found by our method. Distance in the learned embedded space is shown bottom right. Matches are based on the location of the hands, and more generally body pose - not the individual or the background.

estimating body pose by localizing the hands using a parametric, nonlinear multi-layered embedding of the raw pixel images. Unlike many other metric learning approaches, ours is designed for use with real-world images, having a convolutional architecture that scales gracefully to large images and is invariant to local geometric distortions.

Our embedding, trained on both real and synthetic data, is a functional mapping that projects images with similar head and hand positions to lie close-by in a low-dimensional output space. Efficient nearest-neighbour search can then be performed in this space to find images in a large training corpus that have similar pose. Specifically for this task, we have designed an interface to obtain and verify head and hand labels for thousands of frames through Amazon Mechanical Turk with minimal user intervention. We find that our method is able to cope with the terse and noisy labels provided by crowd-sourcing. It succeeds in generalizing to body and hand pose when such cues are not explicitly provided in the labels (see Fig. 1).

## 2 Related work

Our application domain is related to several approaches in the computer vision literature that propose hand or body pose tracking. Many techniques rely on sliding-window part detectors based on color and other features applied to controlled recording conditions ([10, 39, 4, 28] to name a few, we refer to [32] for a complete survey). In our domain, hands might only occupy a few pixels, and the only body-part that can reliably be detected is the human face ([26, 13]). Many techniques have been proposed that extract, learn, or reason over entire body features. Some use a combination of local detectors and structural reasoning (see [33] for coarse tracking and [8] for person-dependent tracking). In a similar spirit, more general techniques using pictorial structures [3, 12, 35], "poselets" [6], and other part-models [11] have received increased attention. An entire new stream of kinematic model-based techniques based on the HumanEva dataset has been proposed [37], but this area differs from our domain in that the images considered are of higher quality and less cluttered.

More closely related to our task are nearest-neighbour and locally-weighted regression-based techniques. Some extract "shape-context" edge based histograms from the human body [25, 1] or just silhouette features [15]. Shakhnarovich et al. [36] use HOG [9] features and boosting for learning a parameter sensitive hash function. All these approaches rely on good background subtraction or recordings with clear backgrounds. Our domain contains clutter, lighting variations and low resolution such that it is impossible to separate body features from background successfully. We instead learn relevant features directly from pixels (instead of pre-coded edge or gradient histogram features), and discover implicitly background invariance from training data.

Several other works [36, 9, 4, 15] have used synthetically created data as a training set. We show in this paper several experiments with challenging real video (with crowd-sourced Amazon Mechanical Turk labels), synthetic training data, and hybrid datasets. Our final system (after training) is always applied to the cluttered non-background subtracted real video input without any labels.

Our technique is also related to distance metric learning, an important area of machine learning research, especially due to recent interest in analyzing complex high-dimensional data. A subset of approaches for dimensionality reduction [17, 16] implicitly learn a distance metric by learning a function (mapping) from high-dimensional (i.e. pixel) space to low-dimensional "feature" space such that perceptually similar observations are mapped to nearby points on a manifold. Neighbourhood Components Analysis (NCA) [14] proposes a solution where the transformation from input to feature space is linear and the distance metric is Euclidean. NCA learns the transformation that is optimal for performing KNN in the feature space. NCA has also been recently extended to the nonlinear case [34] using MNIST class labels and to linear 1D regression for reinforcement learning [20]. Dimensionality Reduction by Learning an Invariant Mapping (DrLIM) [16] also learns a nonlinear mapping. Like NCA, DrLIM uses class neighbourhood structure to drive the optimization: observations with the same class label are driven to be close-by in feature space. Our approach is also inspired by recent hashing methods [2, 34, 38], although those techniques are restricted to binary codes for fast lookup.

## 3 Learning an invariant mapping by nonlinear embedding

We first discuss Neighbourhood Components Analysis [14] and its nonlinear variants. We then propose an alternative objective function optimized for performing nearest neighbour (NN) regression rather than classification. Next, we describe our convolutional architecture which maps images from

high-dimensional to low-dimensional space. Finally we introduce a related but different objective for our model based on DrLIM.

## 3.1 Neighbourhood Components Analysis

NCA (both linear and nonlinear) and DrLIM do not presuppose the existence of a meaningful and computable distance metric in the input space. They only require that neighbourhood relationships be defined between training samples. This is well-suited for learning a metric for non-parametric classification (e.g. KNN) on high-dimensional data. If the original data does not contain discrete class labels, but real-valued labels (e.g. pose information for images of people) one alternative is to define neighbourhoods based on the distance in the real-valued label space and proceed as usual. However, if classification is not our ultimate goal, we may wish to exploit the "soft" nature of the labels and use an alternative objective (i.e. one that does not optimize KNN performance).

Suppose we are given a set of $N$ labeled training cases $\{\mathbf{x}_i, \mathbf{y}_i\}$, $i = 1, 2, \ldots, N$, where $\mathbf{x}_i \in R^D$, and $\mathbf{y}_i \in R^L$. Each training point, $i$, selects another point, $j$, as its neighbour with some probability defined by normalizing distances in the transformed feature space [14]:

$$p_{ij} = \frac{\exp(-d_{ij}^2)}{\sum_{k \neq i} \exp(-d_{ik}^2)}, \quad p_{ii} = 0, \quad d_{ij} = ||\mathbf{z}_i - \mathbf{z}_j||_2 \tag{1}$$

where we use a Euclidean distance metric $d_{ij}$ and $\mathbf{z}_i = f(\mathbf{x}_i|\theta)$ is the mapping (parametrized by $\theta$) from input space to feature space. For NCA this is typically linear, but it can be extended to be nonlinear through back-propagation (for example in [34] it is a multi-layer neural network). NCA assumes that the labels, $\mathbf{y}_i$, are discrete $y_i \in 1, 2, \ldots, C$ rather than real-valued and seeks to maximize the expected number of correctly classified points on the training data which minimizes:

$$L_{\text{NCA}} = -\sum_{i=1}^{N} \sum_{j:y_i=y_j} p_{ij}. \tag{2}$$

The parameters are found by minimizing $L_{\text{NCA}}$ with respect to $\theta$; back-propagating in the case of a multi-layer parametrization. Instead of seeking to optimize KNN classification performance, we can use the NCA regression (NCAR) objective [20]:

$$L_{\text{NCAR}} = \sum_{i=1}^{N} \sum_{j \neq i} p_{ij} ||\mathbf{y}_i - \mathbf{y}_j||_2^2. \tag{3}$$

Intuitively, this states that if, with high probability, $i$ and $j$ are neighbours in feature space, then they should also lie close-by in label space. While we use the Euclidean distance in label space, our approach generalizes to other metrics which may be more appropriate for a different domain.

Keller et al. [20] consider the linear case of NCAR, where $\theta$ is a weight matrix and $y$ is a scalar representing Bellman error to map states with similar Bellman errors close together. Similar to NCA, we can extend this objective to the nonlinear, multi-layer case. We simply need to compute the derivative of $L_{\text{NCAR}}$ with respect to the output of the mapping, $\mathbf{z}_i$, and backpropagate through the remaining layers of the network. The gradient can be computed efficiently as:

$$\frac{\partial L_{\text{NCAR}}}{\partial \mathbf{z}_i} = -2 \sum_{j \neq i} (\mathbf{z}_i - \mathbf{z}_j) \left[ p_{ij} \left( y_{ij}^2 - \delta_i \right) + p_{ji} \left( y_{ij}^2 - \delta_j \right) \right]. \tag{4}$$

where we use $y_{ij}^2 = ||\mathbf{y}_i - \mathbf{y}_j||_2^2$ and $\delta_i = \sum_j p_{ij} y_{ij}^2$. See the supplementary material for details.

## 3.2 Convolutional architectures

As [34] points out, nonlinear NCA was originally proposed in [14] but with the exception of a modest success with a two-layer network in extracting 2D codes that explicitly represented the size and orientation of face images, attempts to extract more complex properties using multi-layer feature extraction were less successful. This was due, in part, to the difficulty in training multi-layer networks and the fact that many data pairs are required to fit the large number of network parameters.

Though both [34] and [38] were successful in learning a multi-layer nonlinear mapping of the data, there is still a fundamental limitation of using fully-connected networks that must be addressed. Such an architecture can only be applied to relatively small image patches (typically less than 64 × 64 pixels), because they do not scale well with the size of the input. Salakhutdinov and Hinton

escaped this issue by training only on the MNIST dataset ($28 \times 28$ images of digits) and Torralba et al. used a global image descriptor [29] as an initial feature representation rather than pixels.

However, to avoid such hand-crafted features which may not be suitable for the task, and to scale to realistic sized inputs, models should take advantage of the pictorial nature of the image input. This is addressed by *convolutional architectures* [21], which exploit the fact that salient motifs can appear anywhere in the image. By employing successive stages of weight-sharing and feature-pooling, deep convolutional architectures can achieve stable latent representations at each layer, that preserve locality, provide invariance to small variations of the input, and drastically reduce the number of free parameters.

Our proposed method which we call Convolutional NCA regression (C-NCAR) is based on a standard convolutional architecture [21, 18]: alternating convolution and subsampling layers followed by a single fully-connected layer (see Fig. 2). It differs from typical convolutional nets in the objective function with which it is trained (i.e. minimizing Eq. 3). Because the loss is defined on pairs of examples, we use a siamese network [5]. Pairs of frames are processed by separate networks with equal weights. The loss is then computed on the output of both networks. Hadsell et al. [16] also use a siamese convolutional network with yet a different objective. They use their method for visualization but not any discriminative task. Mobahi et al. [24] have also recently used a convolutional siamese network in which temporal coherence between pairs of frames drives the regularization of the model rather than the objective. More details of training our network are given in Sec. 4.

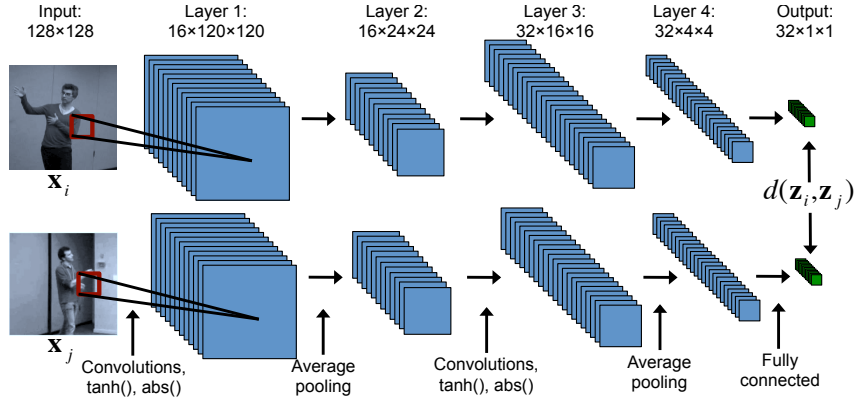

Figure 2: Convolutional NCA regression (C-NCAR). Each image is processed by two convolutional and subsampling layers and one fully-connected layer. A loss (Eq. 3) computed on the distance between resulting codes drives parameter learning.

### 3.3 Adding a contrastive loss function

Like NCA, DrLIM assumes a discrete notion of similarity or dissimilarity between data pairs, $\mathbf{x}_i$ and $\mathbf{x}_j$. It defines both a "similarity" loss, $L_s$, which penalizes similar points which are far apart in code space, and a "dissimilarity" loss, $L_D$, which penalizes dissimilar points which lie within a user-defined margin, $m$, of each other:

$$L_S(\mathbf{x}_i, \mathbf{x}_j) = \frac{1}{2}d_{ij}^2 \qquad L_D(\mathbf{x}_i, \mathbf{x}_j) = \frac{1}{2}\{\max(0, m - d_{ij})\}^2 \qquad (5)$$

where $d_{ij}$ is given by Eq. 1. Let $\gamma_{ij}$ be an indicator such that $\gamma_{ij} = 1$ if $\mathbf{x}_i$ and $\mathbf{x}_j$ are deemed similar and $\gamma_{ij} = 1$ if $\mathbf{x}_i$ and $\mathbf{x}_j$ are deemed dissimilar. For example, if labels $\mathbf{y}_i$ are discrete $y_i \in 1, 2, \ldots, C$, then $\gamma_{ij} = 1$ for $y_i = y_j$ and $\gamma_{ij} = 0$ otherwise. The total loss is defined by:

$$L_{\text{DrLIM}} = \sum_{i=1}^{N} \sum_{j \neq i} \gamma_{ij} L_s(\mathbf{x}_i, \mathbf{x}_j) + (1 - \gamma_{ij}) L_D(\mathbf{x}_i, \mathbf{x}_j). \qquad (6)$$

When faced with real-valued labels, $\mathbf{y}_i$, we can avoid explicitly defining similarity and dissimilarity (e.g. via thresholding) by defining a "soft" notion of similarity:

$$\hat{\gamma}_{ij} = \frac{\exp(-||\mathbf{y}_i - \mathbf{y}_j||_2^2)}{\sum_{k \neq i} \exp(-||\mathbf{y}_i - \mathbf{y}_j||_2^2)}. \qquad (7)$$

Replacing the indicator variables $\gamma_{ij}$ with $\hat{\gamma}_{ij}$ in Eq. 6 yields what we call the *soft* DrLIM loss.

# 4 Experimental results

We evaluate our approach in real and synthetic environments by performing 1-nearest neighbour (NN) regression using a variety of standard and learned metrics described below. For every query image in a test set, we compute its distance (under the metric) to each of the training points in a database. We then copy the label (e.g. (x,y) position of the head and hands) of the neighbour to the query example. For evaluation, we compare the ground-truth label of the query to the label of the nearest neighbour. Errors are reported in terms of mean pixel error over each query and each marker: the head (if it is tracked) and each hand. Errors are absolute with respect to the original image size.

We acknowledge that improved results could potentially be obtained by using more than one neighbour or with more sophisticated techniques such as locally weighted regression [36]. However, we focus on learning a good metric for performing this task rather than the regression problem. The approaches compared are:

**Pixel distance** can be used to find nearest neighbours though it is not practical in real situations due to the intractability of computing distances in such a high-dimensional space.

**GIST** descriptors [29] are a global representation of image content.We are motivated to use GIST by its previous use in nonlinear NCA for image retrieval [38]. The resulting image representation is a length-512 vector. We note that this is still too large for efficient NN search and that the GIST features are not domain-adaptive.

**Linear NCA regression (NCAR)** is described in Section 3. We pre-compute GIST for each image and use that as our input representation. We learn a $512 \times 32$ matrix of weights by minimizing Eq. 3 using nonlinear conjugate gradients with randomly sampled mini-batches of size 512. We perform three line-searches per mini-batch and stop learning after 500 mini-batches. We found that our results slightly improved when we applied a form of local contrast normalization (LCN) prior to computing GIST. Each pixel's response was normalized by the integrated response of a $9 \times 9$ window of neighbouring pixels. For more details see [30].

**Convolutional NCA regression (C-NCAR)** See Fig. 2 for a summary of our architecture. Images are pre-processed using LCN. Convolutions are followed by pixel-wise $\tanh$ and absolute value rectification. The abs prevents cancellations in local neighbourhoods during average downsampling [18]. Our architectural parameters (size of filters, number of filter banks, etc.) are chosen to produce a 32-dimensional output. Derivations of parameter updates are presented as supplementary material.

**Soft DrLIM (S-DrLIM) and Convolutional soft DrLIM (CS-DrLIM)** We also experiment with a variant of an alternative, energy-based method that adds an explicit contrastive loss to the objective rather than implicitly through normalization. The contrastive loss only operates on dissimilar points which lie within a specified margin, $m$, of each other. We use $m = 1.25$ as suggested by [16]. In both the linear and nonlinear case, the architecture and training procedure remains the same as NCAR and C-NCAR, respectively. We use a different objective: minimizing Eq. 6 with respect to the parameters.

## 4.1 Estimating 2D head and hand pose from synthetic data

We extracted 10,000 frames of training data and 5,000 frames of test data from Poser renderings of several hours of real motion capture data. Our synthetic data is similar to that considered in [36], however, we use a variety of backgrounds rather than a constant background. Furthermore, subjects are free to move around the frame and are rendered at various scales. The training set contains 6 different characters superimposed on 9 different backgrounds. The test set contains 6 characters and 8 backgrounds not present in the training set. The inputs, $\mathbf{x}$, are $320 \times 240$ images, and the labels, $\mathbf{y}$, are 6D vectors - the true (x,y) locations of the head and hands.

Results are shown in Table 1 (column SY). Simple linear NCAR performs well compared to the baselines, while our nonlinear methods C-NCAR and CS-DrLIM (which are not restricted to the GIST descriptor) significantly outperform all other approaches. Pixel-based matching (though extremely slow) does surprisingly well. This is perhaps an artifact of the synthetic data.

## 4.2 Estimating 2D hand pose from real video

We digitally recorded all of the contributing and invited speakers at the Learning Workshop (Snowbird) held in April 2010. The set consisted of 30 speakers, with talks ranging from 10-40 minutes each. After each session of talks, blocks of 150 frames were distributed as Human Intelligence Tasks

Table 1: 1-NN regression performance on the synthetic (SY) dataset and the real (RE) dataset. Results are divided into baselines (no learning), linear embeddings and nonlinear embeddings. Errors are the mean pixel distance between the nearest neighbour and the ground truth label of the query. For SY we locate the head and both hands. For RE we assume the location and scale of the head is given by a face detector and only locate the hands. The images at right indicate: (top) a radius of 25.40 pixels with respect to the 320×240 SY input; (bottom) a radius of 16.41 pixels with respect to the 128×128 RE input. Images have been scaled for the plot.

| Embedding | Input | Dim | Error-SY | Error-RE |
|---|---|---|---|---|
| None | Pixels | 16384 | 32.86 | 25.12 |
| None | GIST | 512 | 47.41 | 25.13 |
| PCA | GIST | 128 | 47.17 | 24.85 |
| PCA | GIST | 32 | 48.99 | 25.74 |
| NCAR | GIST | 32 | 34.21 | 24.93 |
| NCAR | LCN+GIST | 32 | 32.90 | 23.15 |
| S-DrLIM | GIST | 32 | 37.80 | 25.19 |
| Boost-SSC [36] | LCN+GIST | 32 | 34.80 | 22.65 |
| C-NCAR | LCN | 32 | 28.95 | **16.41** |
| CS-DrLIM | LCN | 32 | **25.40** | 19.61 |

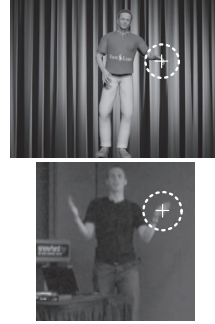

on Amazon Mechanical Turk. We were able to obtain accurate hand and head tracks for each of the speakers within a few hours of their talks. For the following experiments, we divided the 30 speakers into a training set (odd numbered speakers) and test set (even numbered speakers).

Since current state-of-the-art face detection algorithms work reasonably well, we concentrate on the harder problem of tracking the speakers' hands. We first run a commercial face detection algorithm [26] on all frames which provides an estimate of scale for every frame. We use the average scale (per video) estimated by the face detector to crop and rescale each frame to a 128x128 image (centered on the head) that contains the speaker at roughly the same scale as other speakers (there is some variability due to using average scale per video as speakers move throughout their talks). A similar preprocessing step was used in [12]. We do not consider cases in which the hands lie outside the frame or are occluded. This yields 39,792 and 37,671 training and test images, respectively, containing the head and both hands. Since the images are head-centered, the labels, $\mathbf{y}$, used during training are the 4-dimensional vector containing the relative offset of each hand from the head.

We emphasize that finding the hands is an extremely difficult task (sometimes even for human subjects). Frames are low-resolution (typically the hands are 10-15 pixels in diameter) and contain camera movement as well as frequently poor lighting. While previous work has assumed static backgrounds, we confront the changing backgrounds and aim to learn invariance to both scene and subject identity.

Results are shown in Table 1 (column RE). They are organized into three groups: baselines (high-dimensional), and learning-based methods both linear and nonlinear. The linear methods are able to achieve performance comparable to the baseline with the important attribute that distances are computed in a 32-dimensional space. If the codes are made binary (as in [38]) we could use fast approximate hashing techniques to permit real-time tracking using a database of well over 1 million examples. The nonlinear methods show a dramatic improvement over the linear methods, especially our convolutional architectures which learn features from pixels. Boost-SSC [36] is based on a global representation similar to GIST, and so it is restricted in domain adaptivity. We also investigate the performance of C-NCAR on code size (Fig. 5(a)). Performance is impressive even when the dimension in which we compute distances is reduced from 32 to 2. A visualization of the 2D embedding is shown in Fig. 3.

Fig. 4 shows some examples of nearest-neighbour matches under several different metrics. Most apparent is that our methods, and in particular C-NCAR, develop invariance to background and focus on the subject's pose. Both pixel-based and GIST-batch matching are highly driven by the scene (including lighting and background). Though our method is trained only on the relative positions of the hands from the head, it appears to capture something more substantial about body pose in general. We plan on evaluating this result quantitatively, using synthetic data in which we have access to an articulated skeleton.

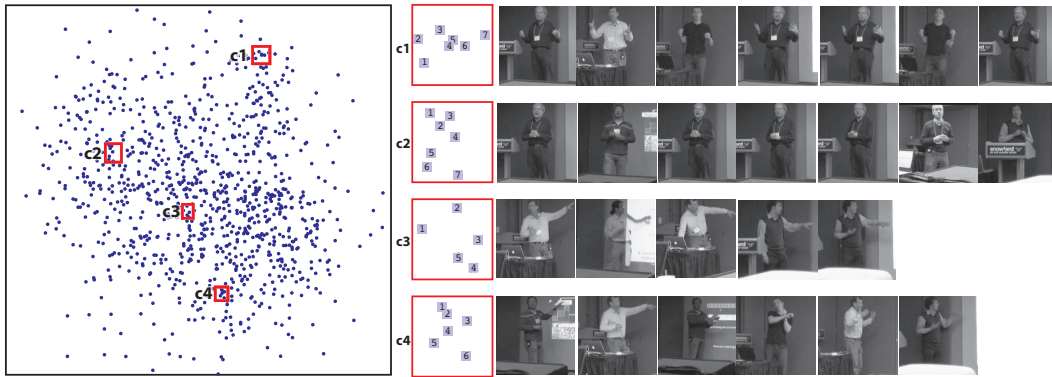

Figure 3: Visualization of the 2D C-NCAR embedding of 1024 points from the RE training set. We show the data points and their local geometry within four example clusters: C1-C4. Note that even with a 2D embedding, we are able to capture pose similarity invariant of subject and background.

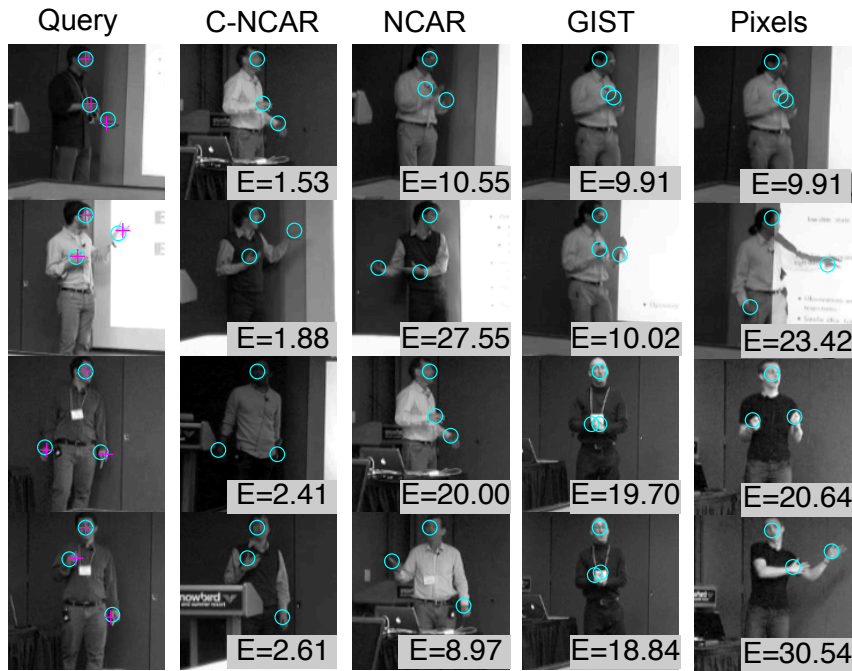

Figure 4: Nearest neighbour pose estimation. The leftmost column shows the query image, and the remaining columns (left to right) show the nearest neighbour found by: nonlinear C-NCAR regression, linear NCAR, GIST, pixel distance. Circles mark the pose obtained by crowd-sourcing; we superimpose the pose estimated by C-NCAR onto the query with crosses.

## 4.3 Improving real-world performance with synthetic data

There has been recent interest in using synthetic examples to improve performance on real-world vision tasks (e.g. [31]). The subtle differences between real and synthetic data make it difficult to apply existing techniques to a dataset comprised of both types of examples. This problem falls under the domain of transfer learning, but to the best of our knowledge, transfer learning between real and synthetic pairings is relatively unexplored. While previous work has attempted to learn representations that are invariant to such effects as geometric distortions of the input [16] and temporal shifts [5, 24] we know of no previous work that has explicitly attempted to learn features that are invariant to the nature of the input, that is, real or synthetic.

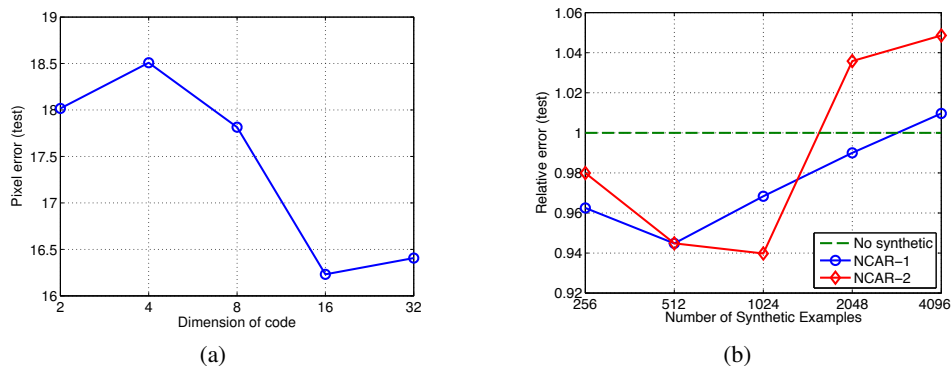

(a)                                        (b)

Figure 5: (a) Effect of code size on the performance of Convolutional NCA regression. (b) Adding synthetic data to a fixed dataset of 1024 real examples to improve test performance measured on real data. Error is expressed relative to a training set with no synthetic data. NCAR-1 does not re-initialize weights when more synthetic examples are added. NCAR-2 reinitializes weights to the same random seed for each run. The curves show that adding synthetic examples improve performance up to a point at which the synthetic examples outnumber the real examples 2:1.

The pairwise nature of our approach is well-suited to learning such invariance, provided that we have established correspondences between real and synthetic examples. In our case of pose estimation, this comes from the labels. By forcing examples with similar poses (regardless of whether they are real or synthetic) to lie close-by in code space we can implicitly produce a representation at each layer that is invariant to the nature of the input. We have not made an attempt to restrict pairings to be only between real and synthetic examples, though this may further aid in learning invariance.

Fig. 5(b) demonstrates the effect of gradually adding synthetic examples from SY to the RE training dataset. We use a reduced-size set of 1024 real examples for training which is gradually modified to contain synthetic examples and a fixed set of 1024 real examples for testing. Error is expressed relative to the case of no synthetic examples. We use Linear NCA for this experiment and train as described above. We follow two different regimes. In NCAR-1 we do not reset the weights of the model to random each time we adjust the training set to add more synthetic examples. We simply add more synthetic data and continue learning. In NCAR-2 we reset the weights to the same random seed for each run. The overall result is the same for each regime: the addition of synthetic examples to the training set improves test performance on real data up to a level at which the number of synthetic examples is double the number of real examples.

## 5 Conclusions

We have presented a nonparametric approach for pose estimation in realistic, challenging video datasets. At the core of our method is a learned parametric mapping from high-dimensional space to a low-dimensional space in which distance is efficiently computed. Our work differs from previous attempts at learning invariant mappings in that it is optimized for nearest neighbour regression rather than classification and it scales to realistic sized images through the use of convolution and weight-sharing. This permits us to learn domain-adaptive features directly from pixels rather than relying on hand-crafted features or global descriptors.

In our experiments, we have restricted ourselves to 1-NN matching, but we plan to investigate other more sophisticated approaches such as locally weighted regression, or using the match as an initialization for a gradient descent search in a parametric model. Though we work with video, our model does not rely on any type of temporal coherence. Integrating temporal knowledge in the form of a prior would benefit our approach. Alternatively, temporal context could be integrated at the input level, from simple frame differencing to more sophisticated temporal feature extraction (e.g. [23]).

Our entire network is trained end-to-end with a single objective, and we do not perform any network pre-training as in [34, 38]. Recent work has demonstrated that pre-training can successfully be applied to convolutional architectures, both in the context of RBMs [22, 27] and sparse coding [19]. We intend to investigate the effect of pre-training, as well as the use of mixed generative and discriminative objectives.

# References

[1] A. Agarwal, B. Triggs, I. Rhone-Alpes, and F. Montbonnot. Recovering 3D human pose from monocular images. *IEEE Transactions on Pattern Analysis and Machine Intelligence*, 28(1):44–58, 2006.

[2] A. Andoni and P. Indyk. Near-optimal hashing algorithms for approximate nearest neighbor in high dimensions. In *FOCS*, pages 459–468, 2006.

[3] M. Andriluka, S. Roth, and B. Schiele. Pictorial structures revisited: People detection and articulated pose estimation. In *CVPR*, 2009.

[4] V. Athitsos, J. Alon, S. Sclaroff, and G. Kollios. Boostmap: A method for efficient approximate similarity rankings. *CVPR*, 2004.

[5] S. Becker and G. Hinton. Self-organizing neural network that discovers surfaces in random-dot stereograms. *Nature*, 355(6356):161–163, 1992.

[6] L. Bourdev and J. Malik. Poselets: Body part detectors trained using 3d human pose annotations. In *ICCV*, sep 2009.

[7] J. Bouvrie. Notes on convolutional neural networks. Unpublished, 2006.

[8] P. Buehler, A. Zisserman, and M. Everingham. Learning sign language by watching TV (using weakly aligned subtitles). *CVPR*, 2009.

[9] N. Dalal, B. Triggs, and C. Schmid. Human detection using oriented histograms of flow and appearance. *ECCV*, 2006.

[10] A. Farhadi, D. Forsyth, and R. White. Transfer Learning in Sign language. In *CVPR*, 2007.

[11] P. Felzenszwalb, D. McAllester, and D. Ramanan. A discriminatively trained, multiscale, deformable part model. In *CVPR*, 2008.

[12] V. Ferrari, M. Marin-Jimenez, and A. Zisserman. Pose search: Retrieving people using their pose. In *CVPR*, 2009.

[13] A. Frome, G. Cheung, A. Abdulkader, M. Zennaro, B. Wu, A. Bissacco, H. Adam, H. Neven, and L. Vincent. Large-scale Privacy Protection in Google Street View. In *ICCV*, 2009.

[14] J. Goldberger, S. Roweis, G. Hinton, and R. Salakhutdinov. Neighbourhood components analysis. In *NIPS*, 2004.

[15] K. Grauman, G. Shakhnarovich, and T. Darrell. Inferring 3d structure with a statistical image-based shape model. In *ICCV*, pages 641–648, 2003.

[16] R. Hadsell, S. Chopra, and Y. LeCun. Dimensionality reduction by learning an invariant mapping. In *CVPR*, pages 1735–1742, 2006.

[17] G. Hinton and R. Salakhutdinov. Reducing the dimensionality of data with neural networks. *Science*, 313(5786):504 – 507, 2006.

[18] K. Jarrett, K. Kavukcuoglu, M-A Ranzato, and Y. LeCun. What is the best multi-stage architecture for object recognition? In *ICCV*, 2009.

[19] K. Kavukcuoglu, M-A Ranzato, and Y. LeCun. Fast inference in sparse coding algorithms with applications to object recognition. Technical report, NYU, 2008. CBLL-TR-2008-12-01.

[20] P. Keller, S. Mannor, and D. Precup. Automatic basis function construction for approximate dynamic programming and reinforcement learning. In *ICML*, pages 449–456, 2006.

[21] Y. LeCun, L. Bottou, Y. Bengio, and P. Haffner. Gradient-based learning applied to document recognition. *Proc. IEEE*, 86(11):2278–2324, 1998.

[22] H. Lee, R. Grosse, R. Ranganath, and A. Y. Ng. Convolutional deep belief networks for scalable unsupervised learning of hierarchical representations. In *ICML*, pages 609–616, 2009.

[23] R. Memisevic and G. Hinton. Unsupervised learning of image transformations. In *CVPR*, 2007.

[24] H. Mobahi, R. Collobert, and J. Weston. Deep learning from temporal coherence in video. In *ICML*, pages 737–744, 2009.

[25] G. Mori and J. Malik. Estimating human body configurations using shape context matching. *ECCV*, 2002.

[26] M. Nechyba, L. Brandy, and H. Schneiderman. Pittpatt face detection and tracking for the CLEAR 2007 evaluation. *Multimodal Technologies for Perception of Humans*, 2008.

[27] M. Norouzi, M. Ranjbar, and G. Mori. Stacks of convolutional restricted boltzmann machines for shift-invariant feature learning. In *CVPR*, 2009.

[28] S.J. Nowlan and J.C. Platt. A convolutional neural network hand tracker. In *NIPS*, 1995.

[29] A. Oliva and A. Torralba. Modeling the shape of the scene: A holistic representation of the spatial envelope. *International Journal of Computer Vision*, 42(3):145–175, 2001.

[30] N. Pinto, D. Cox, and J. DiCarlo. Why is real-world visual object recognition hard? *PLoS Comput Biol*, 4(1), 2008.

[31] N. Pinto, D. Doukhan, J. DiCarlo, and David D. Cox. A high-throughput screening approach to discovering good forms of biologically inspired visual representation. *PLoS Comput Biol*, 5(11), 11 2009.

[32] R. Poppe. Vision-based human motion analysis: An overview. *Computer Vision and Image Understanding*, 108(1-2):4–18, 2007.

[33] D. Ramanan, D. Forsyth, and A. Zisserman. Strike a pose: Tracking people by finding stylized poses. In *CVPR*, 2005.

[34] R. Salakhutdinov and G. Hinton. Learning a nonlinear embedding by preserving class neighbourhood structure. In *AISTATS*, volume 11, 2007.

[35] B. Sapp, C. Jordan, and B.Taskar. Adaptive pose priors for pictorial structures. In *CVPR*, 2010.

[36] G. Shakhnarovich, P. Viola, and T. Darrell. Fast pose estimation with parameter-sensitive hashing. In *ICCV*, pages 750–759, 2003.

[37] L. Sigal, A. Balan, and Black. M. J. HumanEva: Synchronized video and motion capture dataset and baseline algorithm for evaluation of articulated human motion. *IJCV*, 87(1/2):4–27, 2010.

[38] A. Torralba, R. Fergus, and Y. Weiss. Small codes and large image databases for recognition. In *CVPR*, 2008.

[39] C. Wren, A. Azarbayejani, T. Darrell, and A. Pentland. Pfinder: Real-time tracking of the human body. *IEEE Transactions on Pattern Analysis and Machine Intelligence*, 19(7):780–785, 1997.

